# Multi-step Linear Dyna-style Planning

**Hengshuai Yao**
Department of Computing Science
University of Alberta
Edmonton, AB, Canada T6G2E8

**Shalabh Bhatnagar**
Department of Computer Science
and Automation
Indian Institute of Science
Bangalore, India 560012

**Dongcui Diao**
School of Economics and Management
South China Normal University
Guangzhou, China 518055

## Abstract

In this paper we introduce a multi-step linear Dyna-style planning algorithm. The key element of the multi-step linear Dyna is a multi-step linear model that enables multi-step projection of a sampled feature and multi-step planning based on the simulated multi-step transition experience. We propose two multi-step linear models. The first iterates the one-step linear model, but is generally computationally complex. The second interpolates between the one-step model and the infinite-step model (which turns out to be the LSTD solution), and can be learned efficiently online. Policy evaluation on Boyan Chain shows that multi-step linear Dyna learns a policy faster than single-step linear Dyna, and generally learns faster as the number of projection steps increases. Results on Mountain-car show that multi-step linear Dyna leads to much better online performance than single-step linear Dyna and model-free algorithms; however, the performance of multi-step linear Dyna does not always improve as the number of projection steps increases. Our results also suggest that previous attempts on extending LSTD for online control were unsuccessful because LSTD looks infinite steps into the future, and suffers from the model errors in non-stationary (control) environments.

## 1 Introduction

Linear Dyna-style planning extends Dyna to linear function approximation (Sutton, Szepesvári, Geramifard & Bowling, 2008), and can be used in large-scale applications. However, existing Dyna and linear Dyna-style planning algorithms are all single-step, because they only simulate sampled features one step ahead. This is many times insufficient as one does not exploit in such a case all possible future results. We extend linear Dyna architecture by using a multi-step linear model of the world, which gives what we call the *multi-step linear Dyna-style planning*. Multi-step linear Dyna-style planning is more advantageous than existing linear Dyna, because a multi-step model of the world can project a feature multiple steps into the future and give more steps of results from the feature.

For policy evaluation we introduce two multi-step linear models. The first is generated by iterating the one-step linear model, but is computationally complex when the number of features is large. The second, which we call the $\lambda$-model, interpolates between the one-step linear model and an infinite-step linear model of the world, and is computationally efficient to compute online. Our multi-step linear Dyna-style planning for policy evaluation, Dyna($k$), uses the multi-step linear models to generate $k$-steps-ahead prediction of the sampled feature, and applies a generalized TD (temporal dif-

ference, e.g., see (Sutton & Barto, 1998)) learning on the imaginary multi-step transition experience. When $k$ is equal to 1, we recover the existing linear Dyna-style algorithm; when $k$ goes to infinity, we actually use the LSTD (Bradtke & Barto, 1996; Boyan, 1999) solution for planning.

For the problem of control, related work include least-squares policy iteration (LSPI) (Lagoudakis & Parr, 2001; Lagoudakis & Parr, 2003; Li, Littman & Mansley, 2009), and linear Dyna-style planning for control. LSPI is an offline algorithm, that learns a greedy policy out of a data set of experience, through a number of iterations, each of which sweeps the data set and alternates between LSTD and policy improvement. Sutton et al. (2008) explored the use of linear function approximation with Dyna for control, which does planning using a set of linear action models built from state to state. In this paper, we first build a one-step model from state-action pair to state-action pair through tracking the greedy policy. Using this tracking model for planning is in fact another way of doing single-step linear Dyna-style planning. In a similar manner to policy evaluation, we also have two multi-step models for control. We build the iterated multi-step model by iterating the one-step tracking model. Also, we build a $\lambda$-model for control by interpolating the one-step tracking model and the infinite-step model (also built through tracking). As the infinite-step model coincides with the LSTD solution, we actually propose an online LSTD control algorithm.

Policy evaluation on Boyan Chain shows that multi-step linear Dyna learns a policy faster than single-step linear Dyna. Results on the Mountain-car experiment show that multi-step linear Dyna can find the optimal policy faster than single-step linear Dyna; however, the performance of multi-step linear Dyna does not always improve as the number of projection steps increases. In fact, LSTD control and the infinite-step linear Dyna for control are both unstable, and some intermediate value of $k$ makes the $k$-step linear Dyna for control perform the best.

## 2 Backgrounds

Given a Markov decision process (MDP) with a state space $\mathbb{S} = \{1, 2, \ldots, N\}$, the problem of policy evaluation is to predict the long-term reward of a policy $\pi$ for every state $s \in \mathbb{S}$:

$$V^\pi(s) = \sum_{t=0}^{\infty} \gamma^t r_t, \quad 0 < \gamma < 1, \quad s_0 = s,$$

where $r_t$ is the reward received by the agent at time $t$. Given $n$ ($n \le N$) feature functions $\varphi_j : \mathbb{S} \mapsto \mathbb{R}$, $j = 1, \ldots, n$, the feature of state $i$ is $\phi(i) = [\varphi_1(i), \varphi_2(i), \ldots, \varphi_n(i)]^T$. Now $V^\pi$ can be approximated using $\hat{V}^\pi = \Phi\theta$, where $\theta$ is the weight vector, and $\Phi$ is the feature matrix whose entries are $\Phi_{i,j} = \varphi_j(i)$, $i = 1, \ldots, N$; $j = 1, \ldots, n$. At time $t$, linear TD(0) updates the weights as

$$\theta_{t+1} = \theta_t + \alpha_t \delta_t \phi_t, \quad \delta_t = r_t + \gamma \theta_t^T \phi_{t+1} - \theta_t^T \phi_t,$$

where $\alpha_t$ is a positive step-size and $\phi_t$ corresponds to $\phi(s_t)$.

Most of earlier work on Dyna uses a lookup table representation of states (Sutton, 1990; Sutton & Barto, 1998). Modern Dyna is more advantageous in the use of linear function approximation, which is called *linear Dyna-style planning* (Sutton et al., 2008). We denote the state transition probability matrix of policy $\pi$ by $P^\pi$, whose $(i, j)$th component is $P_{i,j}^\pi = E_\pi\{s_{t+1} = j | s_t = i\}$; and denote the expected reward vector of policy $\pi$ by $R^\pi$, whose $i$th component is the expected reward of leaving state $i$ in one step. Linear Dyna tries to estimate a compressed model of policy $\pi$:

$$(F^\pi)^T = (\Phi^T D^\pi \Phi)^{-1} \cdot \Phi^T D^\pi P^\pi \Phi; \qquad f^\pi = (\Phi^T D^\pi \Phi)^{-1} \cdot \Phi^T D^\pi R^\pi,$$

where $D^\pi$ is the $N \times N$ matrix whose diagonal entries correspond to the steady distribution of states under policy $\pi$. $F^\pi$ and $f^\pi$ constitute the world model of linear Dyna for policy evaluation, and are estimated online through gradient descent:

$$F_{t+1}^\pi = F_t^\pi + \beta_t(\phi_{t+1} - F_t^\pi \phi_t)\phi_t^T; \qquad f_{t+1}^\pi = f_t^\pi + \beta_t(r_t - \phi_t^T f_t^\pi)\phi_t, \qquad (1)$$

respectively, where the features and reward are all from real world experience and $\beta_t$ is the modeling step-size.

Dyna repeats some steps of planning in each of which it samples a feature, projects it using the world model, and plans using linear TD(0) based on the imaginary experience. For policy evaluation, the

fixed-point of linear Dyna is the same as that of linear TD(0) under some assumptions (Tsitsiklis & Van Roy, 1997; Sutton et al., 2008), that satisfies

$$A^\pi \theta^* + b^\pi = 0: \quad A^\pi = \Phi^T D^\pi (\gamma P^\pi - I)\Phi; \qquad b^\pi = \Phi^T D^\pi R^\pi,$$

where $I_{N \times N}$ is the identity matrix.

## 3 The Multi-step Linear Model

In the lookup table representation, $(P^\pi)^T$ and $R^\pi$ constitute the one-step world model. The $k$-step transition model of the world is obtained by iterating $(P^\pi)^T$, $k$ times with discount (Sutton, 1995):

$$P^{(k)} = (\gamma (P^\pi)^T)^k, \quad \forall k = 1, 2, \dots$$

At the same time we accumulate the rewards generated in the process of this iterating:

$$R^{(k)} = \sum_{j=0}^{k-1} (\gamma P^\pi)^j R^\pi, \quad \forall k = 1, 2, \dots,$$

where $R^{(k)}$ is called the $k$-step reward model. $P^{(k)}$ and $R^{(k)}$ predict a feature $k$ steps into the future. In particular, $P^{(k)}\phi$ is the feature of the expected state after $k$ steps from $\phi$, and $(R^{(k)})^T \phi$ is the expected accumulated rewards in $k$ steps from $\phi$. Notice that

$$V^\pi = R^{(k)} + (P^{(k)})^T V^\pi, \quad \forall k = 1, 2, \dots, \tag{2}$$

which is a generalization of the Bellman equation, $V^\pi = R^\pi + \gamma P^\pi V^\pi$.

### 3.1 The Iterated Multi-step Linear Model

In the linear function approximation, $F^\pi$ and $f^\pi$ constitute the one-step linear model. Similar to the lookup table representation, we can iterate $F^\pi$, $k$ times, and accumulate the approximated rewards along the way:

$$F^{(k)} = (\gamma F^\pi)^k; \qquad f^{(k)} = \sum_{j=0}^{k-1} (\gamma (F^\pi)^T)^j f^\pi.$$

We call $(F^{(k)}, f^{(k)})$ the *iterated* multi-step linear model. By this definition, we extend (2) to the $k$-step linear Bellman equation:

$$\hat{V}^\pi = \Phi\theta^* = \Phi f^{(k)} + \Phi(F^{(k)})^T \theta^*, \quad \forall k = 1, 2, \dots, \tag{3}$$

where $\theta^*$ is the linear TD(0) solution.

### 3.2 The $\lambda$-model

The quantities $F^{(k)}$ and $f^{(k)}$ require powers of $F^\pi$. One can first estimate $F^\pi$ and $f^\pi$, and then estimate $F^{(k)}$ and $f^{(k)}$ using powers of the estimated $F^\pi$. However, real life tasks require a lot of features. Generally $(F^\pi)^k$ requires $O((k-1)n^3)$ computation, which is too complex when the number of features ($n$) is large.

Rather than using $F^{(k)}$ and $f^{(k)}$, we would like to explore some other multi-step model that is cheap in computation but is still meaningful in some sense. First let us see how $F^{(k)}$ and $f^{(k)}$ are used if they can be computed. Given an imaginary feature $\tilde{\phi}_\tau$, we look $k$ steps ahead to see our future feature by applying $F^{(k)}$:

$$\tilde{\phi}_\tau^{(k)} = F^{(k)} \tilde{\phi}_\tau.$$

As $k$ grows, $F^{(k)}$ diminishes and thus $\tilde{\phi}_\tau^{(k)}$ converges to 0. [1] This means that the more steps we look into the future from a given feature, the more ambiguous is our resulting feature. It suggests that we

can use a decayed one-step linear model to approximate the effects of looking multiple steps into the future:
$$L^{(k)} = (\lambda\gamma)^{k-1}\gamma F^\pi,$$
parameterized by a factor $\lambda \in (0, 1]$. To guarantee that the optimality (3) still holds, we define
$$l^{(k)} = (I - (L^{(k)})^T)(I - \gamma(F^\pi)^T)^{-1}f^\pi.$$
We call $(L^{(k)}, l^{(k)})$ the $\lambda$-*model*. When $k = 1$, we have $L^{(1)} = F^{(1)} = \gamma F^\pi$ and $l^{(1)} = f^{(1)} = f^\pi$, recovering the one-step model used by existing linear Dyna. Notice that $L^{(k)}$ diminishes as $k$ grows, which is consistent with the fact that $F^{(k)}$ also diminishes as $k$ grows. Finally, the infinite-step model reduces to a single vector, $l^{(\infty)} = f^{(\infty)} = \theta^*$. The intermediate $k$ interpolates between the single-step model and infinite-step model.

For intermediate $k$, computation of $L^{(k)}$ has the same complexity as the estimation of $F^\pi$. Interestingly, all $l^{(k)}$ can be obtained by shifting from $l^{(\infty)}$ by an amount that shrinks $l^{(\infty)}$ itself: [2]
$$
\begin{aligned}
l^{(k)} &= (I - (L^{(k)})^T)(I - \gamma(F^\pi)^T)^{-1}f^\pi, \\
&= l^{(\infty)} - (L^{(k)})^T l^{(\infty)}.
\end{aligned}
\tag{4}
$$
The case of $k = 1$ is interesting. The linear Dyna algorithm (Sutton et al., 2008) takes advantage of the fact that $l^{(1)} = f^\pi$ and estimates it through gradient descent. On the other hand, in our Dyna algorithm, we use (4) and estimate all $l^{(k)}$ from the estimation of $l^{(\infty)}$, which is generally no longer a gradient-descent estimate.

## 4 Multi-step Linear Dyna-style Planning for Policy Evaluation

The architecture of multi-step linear Dyna-style planning, Dyna($k$), is shown in Algorithm 1. Generally any valid multi-step model can be used in the architecture. For example, in the algorithm we can take $M^{(k)} = F^{(k)}$ and $m^{(k)} = f^{(k)}$, giving us a linear Dyna architecture using the iterated multi-step linear model, which we call the *Dyna(k)-iterate*.

In the following we present the family of Dyna($k$) planning algorithms that use the $\lambda$-model. We first develop a planning algorithm for the infinite-step model, and based on it we then present Dyna($k$) planning using the $\lambda$-model for any finite $k$.

### 4.1 Dyna($\infty$): Planning using the Infinite-step Model

The infinite-step model is preferable in computation because $F^{(\infty)}$ diminishes and the model reduces to $f^{(\infty)}$. It turns out that $f^{(\infty)}$ can be further simplified to allow an efficient online estimation:
$$
\begin{aligned}
f^{(\infty)} &= (I - \gamma(F^\pi)^T)^{-1}f^\pi \\
&= (\Phi^T D^\pi \Phi - \gamma\Phi^T D^\pi P^\pi \Phi)^{-1} \cdot \Phi^T D^\pi \Phi f^\pi \\
&= -(A^\pi)^{-1}b^\pi.
\end{aligned}
$$
We can accumulate $A^\pi$ and $b^\pi$ online like LSTD (Bradtke & Barto, 1996; Boyan, 1999; Xu et al., 2002) and solve $f^{(\infty)}$ by matrix inversion methods or recursive least-square methods.

As with traditional Dyna, we initially sample a feature $\tilde\phi$ from some distribution $\mu$. We then apply the infinite-step model to get the expected future features and all the possible future rewards:
$$\tilde\phi^{(\infty)} = F^{(\infty)}\tilde\phi; \qquad \tilde r^{(\infty)} = (f^{(\infty)})^T\tilde\phi.$$
Next, a generalized linear TD(0) is applied on this simulated experience.
$$\tilde\theta := \tilde\theta + \alpha(\tilde r^{(\infty)} + \tilde\theta^T\tilde\phi^{(\infty)} - \tilde\theta^T\tilde\phi)\tilde\phi.$$
Because $\tilde\phi^{(\infty)} = 0$, this simplifies into
$$\tilde\theta := \tilde\theta + \alpha(\tilde r^{(\infty)} - \tilde\theta^T\tilde\phi)\tilde\phi.$$
We call this algorithm Dyna($\infty$), which actually uses the LSTD solution for planning.

**Algorithm 1** Dyna($k$) algorithm for evaluating policy $\pi$ (using any valid $k$-step model).

---
Initialize $\theta_0$ and some model
Select an initial state
**for** each time step **do**
    Take an action $a$ according to $\pi$, observing $r_t$ and $\phi_{t+1}$
    $\theta_{t+1} = \theta_t + \alpha_t(r_t + \gamma\phi_{t+1}^T\theta_t - \phi_t^T\theta_t)\phi_t$           /* linear TD(0) */
    Update $M^{(k)}$ and $m^{(k)}$
    Set $\tilde{\theta}_0 = \theta_{t+1}$
    **repeat** $\tau = 1$ **to** $p$                              /*Planning*/
        Sample $\tilde{\phi}_\tau \sim \mu(\cdot)$
        $\tilde{\phi}^{(k)} = M^{(k)}\tilde{\phi}_\tau$                    /* $\tilde{\phi}^{(\infty)} = 0$*/
        $\tilde{r}^{(k)} = (m^{(k)})^T\tilde{\phi}_\tau$
        $\tilde{\theta}_{\tau+1} := \tilde{\theta}_\tau + \alpha_\tau(\tilde{r}_\tau^{(k)} + \tilde{\theta}_\tau^T\tilde{\phi}_\tau^{(k)} - \tilde{\theta}_\tau^T\tilde{\phi}_\tau)\tilde{\phi}_\tau$    /*Generalized $k$-step linear TD(0) learning */
    Set $\theta_{t+1} = \tilde{\theta}_{\tau+1}$
**end for**

---

## 4.2 Planning using the $\lambda$-model

The $k$-step $\lambda$-model is efficient to estimate, and can be directly derived from the single-step and infinite-step models:

$$L^{(k)} = (\lambda\gamma)^{k-1}\gamma F_{t+1}^\pi; \quad l^{(k)} = f^{(\infty)} - (L^{(k)})^T f^{(\infty)},$$

respectively, where the infinite-step model is estimated by $f^{(\infty)} = (A_{t+1}^\pi)^{-1}b_{t+1}^\pi$. Given an imaginary feature $\tilde{\phi}$, we look $k$ steps ahead to see the future features and rewards:

$$\tilde{\phi}^{(k)} = L^{(k)}\tilde{\phi}; \qquad \tilde{r}^{(k)} = (l^{(k)})^T\tilde{\phi}.$$

Thus we obtain an imaginary $k$-step transition experience $\tilde{\phi} \to (\tilde{\phi}^{(k)}, \quad \tilde{r}^{(k)})$, on which we apply a $k$-step version of linear TD(0):

$$\tilde{\theta}_{\tau+1} = \tilde{\theta}_\tau + \alpha(\tilde{r}^{(k)} + \tilde{\theta}_\tau^T\tilde{\phi}^{(k)} - \tilde{\theta}_\tau^T\tilde{\phi})\tilde{\phi}.$$

We call this algorithm the *Dyna(k)-lambda* planning algorithm. When $k = 1$, we obtain another single-step Dyna, Dyna(1). Notice that Dyna(1) uses $f^{(\infty)}$ while the linear Dyna uses $f^\pi$. When $k \to \infty$, we obtain the Dyna($\infty$) algorithm.

## 5 Planning for Control

Planning for control is more difficult than that for policy evaluation because in control the policy changes from time step to time step. Linear Dyna uses a separate model for each action, and these action models are from state to state (Sutton et al., 2008). Our model for control is different in that it is from state-action pair to state-action pair. However, rather than building a model for all state-action pairs, we build only one state-action model that tracks the sequence of greedy actions. Using this greedy-tracking model is another way of doing linear Dyna-style planning. In the following we first build the single-step greedy-tracking model and the infinite-step greedy-tracking model, and based on these tracking models we build the iterated model and the $\lambda$-model.

Our extension of linear Dyna to control contains a TD control step (we use Q-learning), and we call it the linear Dyna-Q architecture. In the Q-learning step, the next feature is already implicitly selected. Recall that Q-learning selects the largest next $Q$-function as the target for TD learning, which is $\max_{\mathbf{a}'}\hat{Q}_{t+1}(s_{t+1},\mathbf{a}') = \max_{\mathbf{a}'}\phi(s_{t+1},\mathbf{a}')^T\theta_t$. Alternatively, the greedy next state-action feature

$$\vec{\phi}_{t+1} = \arg\max_{\phi'=\phi(s_{t+1},\cdot)} \phi'^T\theta_t$$

is selected by Q-learning. We build a single-step projection matrix between state-action pairs, $F$, by moving its projection of the current feature towards the greedy next state-action feature (tracking):

$$F_{t+1} = F_t + \beta_t(\vec{\phi}_{t+1} - F_t\phi_t)\phi_t^T. \tag{5}$$

**Algorithm 2** Dyna-Q($k$)-lambda: $k$-step linear Dyna-Q algorithm for control (using the $\lambda$-model).

Initialize $F_0$, $A_0$, $b_0$ and $\theta_0$
Select an initial state
**for** each time step **do**
    Take action $a$ at $s_t$ (using $\epsilon$-greedy), observing $r_t$ and $s_{t+1}$
    Choose $a'$ that leads to the largest $\hat{Q}(s_{t+1}, \boldsymbol{a}')$
    Set $\phi = \phi(s_t, a)$, $\vec{\phi} = \phi(s_{t+1}, a')$
    $\theta_{t+1} = \theta_t + \alpha_t(r_t + \gamma\vec{\phi}^T\theta_t - \phi^T\theta_t)\phi$         /*Q-learning*/
    $A_{t+1} = A_t + \phi_t(\gamma\vec{\phi}^T - \phi)^T$,    $b_{t+1} = b_t + \phi_t r_t$
    $f^{(\infty)} = -(A_{t+1})^{-1}b_{t+1}$        /*Using matrix inversion or recursive least-squares */
    $F_{t+1} = F_t + \alpha_t(\vec{\phi} - F_t\phi)\phi^T$,
    $L^{(k)} = (\lambda\gamma)^{k-1}\gamma F_{t+1}$
    $l^{(k)} = f^{(\infty)} - (L^{(k)})^T f^{(\infty)}$
    Set $\tilde{\theta}_0 = \theta_{t+1}$
    **repeat $\tau$ times**                     /*Planning*/
        Sample $\tilde{\phi}_\tau \sim \mu$
        $\tilde{\phi}^{(k)} = L^{(k)}\tilde{\phi}_\tau$
        $\tilde{r}^{(k)} = (l^{(k)})^T\tilde{\phi}_\tau$
        $\tilde{\theta}_{\tau+1} := \tilde{\theta}_\tau + \alpha_\tau(\tilde{r}^{(k)}_\tau + \tilde{\theta}^T_\tau\tilde{\phi}^{(k)} - \tilde{\theta}^T_\tau\tilde{\phi}_\tau)\tilde{\phi}_\tau$
    Set $\theta_{t+1} = \tilde{\theta}_{\tau+1}$
**end for**

Estimation of the single-step reward model, $f$, is the same as in policy evaluation.

In a similar manner, in the infinite-step model, matrix $A$ is updated using the greedy next feature, while vector $b$ is updated in the same way as in LSTD. Given $A$ and $b$, we can solve them and get $f^{(\infty)}$. Once the one-step model and the infinite-step model are available, we interpolate them and compute the $\lambda$-model in a similar manner to policy evaluation. The complete multi-step Dyna-Q control algorithm using the $\lambda$-model is shown in Algorithm 2. We noticed that $f^{(\infty)}$ can be directly used for control, giving an online LSTD control algorithm.

We can also extend the iterated multi-step model and Dyna($k$)-iterate to control. Given the single-step greedy-tracking model, we can iterate it and get the iterated multi-step linear model in a similar way to policy evaluation. The linear Dyna for control using the iterated greedy-tracking model (which we call Dyna-Q(k)-iterate) is straightforward and thus not shown.

# 6 Experimental Results

## 6.1 Boyan Chain Example

The problem we consider is exactly the same as that considered by Boyan (1999). The root mean square error (RMSE) of the value function is used as a criterion. Previously it was shown that linear Dyna can learn a policy faster than model-free TD methods in the beginning episodes (Sutton et al., 2008). However, after some episodes, their implementation of linear Dyna became poorer than TD. A possible reason leading to their results may be that the step-sizes of learning, modeling and planning were set to the same value. Also, their step-size diminishes according to $1/(traj\#)^{1.1}$, which does not satisfy the standard step-size rule required for stochastic approximation. In our linear Dyna algorithms, we used different step-sizes for learning, modeling and planning.

(1) *Learning step-size*. We used here the same step-size rule for TD as Boyan (1999), where $\alpha = 0.1(1 + 100)/(traj\# + 100)$ was found to be the best in the class of step-sizes and also used it for TD in the learning sub-procedure of all linear Dyna algorithms. (2) *Modeling step-size*. For Dyna($k$)-lambda, we used $\beta_T = 0.5(1 + 10)/(10 + T)$ for estimation of $F^\pi$, where $T$ is the number of state visits across episodes. For linear Dyna, the estimation of $F^\pi$ and $f^\pi$ also used the same $\beta_T$. (3) *Planning step-size*. In our experiments all linear Dyna algorithms simply used $\alpha_\tau = 0.1$.

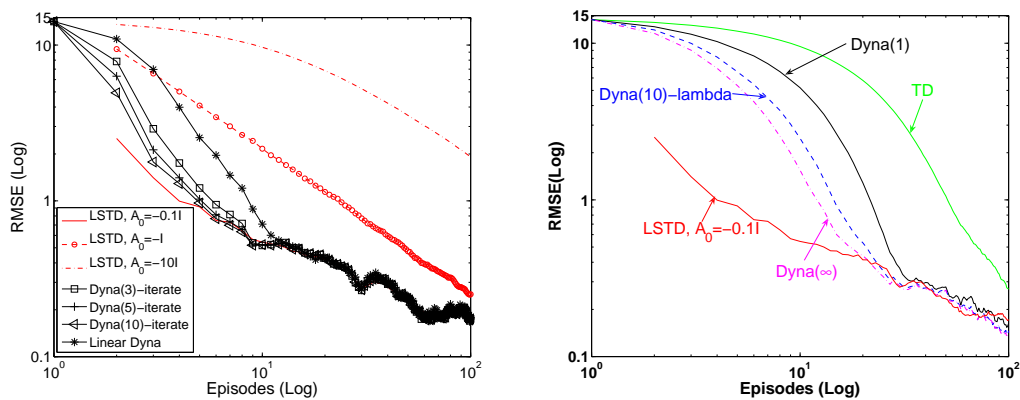

Figure 1: Results on Boyan Chain. Left: comparison of RMSE of Dyna($k$)-iterate with LSTD. Right: comparison of RMSE of Dyna($k$)-lambda with TD and LSTD.

The weights of various learning algorithms, $f^\pi$ for the linear Dyna, and $b^\pi$ for Dyna($k$) were all initialized to zero. No eligibility trace is used for any algorithm. In the planning step, all Dyna algorithms sampled a unit basis vector whose nonzero component was in a uniformly random location. In the following we report the results of planning only once. All RMSEs of algorithms were averaged over 30 (identical) sets of trajectories.

Figure 1 (left) shows the performance of Dyna($k$)-iterate and LSTD, and Figure 1 (right) shows the performance of Dyna($k$)-lambda, LSTD and TD. All linear Dyna algorithms were found to be significantly and consistently faster than TD. Furthermore, multi-step linear Dyna algorithms were much faster than single-step linear Dyna algorithms. Matrix $A$ of LSTD and Dyna($k$)-lambda needs perturbation in initialization, which has a great impact on the performances of two algorithms. For LSTD, we tried initialization of $A_0^\pi$ to $-10I, -I, -0.1I$, and showed their effects in Figure 1 (left), in which $A_0^\pi = -0.1I$ was the best for LSTD. Similar to LSTD, Dyna($k$)-lambda is also sensitive to $A_0^\pi$. Linear Dyna and Dyna($k$)-iterate do not use $A^\pi$ and thus do not have to tune $A_0^\pi$. $F^\pi$ was initialized to 0 for Dyna($k$) ($k < \infty$) and linear Dyna. In Figure 1 (right) LSTD and Dyna($k$)-lambda were compared under the same setting (Dyna($k$)-lambda also used $A_0 = -0.1I$). Dyna($k$)-lambda used $\lambda = 0.9$.

## 6.2 Mountain-car

We used the same Mountain-car environment and tile coding as in the linear Dyna paper (Sutton et al., 2008). The state feature has a dimension of $10,000$. The state-action feature is shifted from the state feature, and has a dimension of $30,000$ because there are three actions of the car. Because the feature and matrix are really large, we were not able to compute the iterated model, and hence we only present here the results of Dyna-Q($k$)-lambda.

Experimental setting. (1)*Step-sizes*. The Q-learning step-size was chosen to be $0.1$, in both the independent algorithm and the sub-procedure of Dyna-Q($k$)-lambda. The planning step-size was $0.1$. The matrix $F$ is much more dense than $A$ and leads to a very slow online performance. To tackle this problem, we avoided computing $F$ explicitly, and used a least-squares computation of the projection, given in the supplementary material. In this implementation, there is no modeling step-size. (2)*Initialization*. The parameters $\theta$ and $b$ were both initialized to 0. $A$ was initialized to $-I$. (3)*Other setting*. The $\lambda$ value for Dyna-Q($k$)-lambda was $0.9$. We recorded the state-action pairs online and replayed the feature of a past state-action pair in planning. We also compared the linear Dyna-style planning for control (with state features) (Sutton et al., 2008), which has three sets of action models for this problem. In linear Dyna-style planning for control we replayed a state feature of a past time step, and projected it using the model of the action that was selected at that time step. No eligibility trace or exploration was used. Results reported below were all averaged over 30 independent runs, each of which contains 20 episodes.

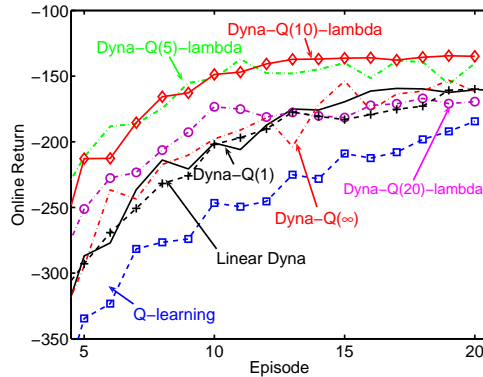

Figure 2: Results on Mountain-car: comparison of online return of Dyna-Q($k$)-lambda, Q-learning and linear Dyna for control.

Results are shown in Figure 2. Linear Dyna-style planning algorithms were found to be significantly faster than Q-learning. Multi-step planning algorithms can be still faster than single-step planning algorithms. The results also show that planning too many steps into the future is harmful, *e.g.*, Dyna-Q(20)-lambda and Dyna-Q($\infty$) gave poorer performance than Dyna-Q(5)-lambda and Dyna-Q(10)-lambda. This shows that some intermediate values of $k$ trade off the model accuracy and the depth of looking ahead, and performed best. In fact, Dyna-Q($\infty$) and LSTD control algorithm were both unstable, and typically failed once or twice in 30 runs. The intuition is that in control the policy changes from time step to time step and the model is highly non-stationary. By solving the model and looking infinite steps into the future, LSTD and Dyna-Q($\infty$) magnify the errors in the model.

## 7 Conclusion and Future Work

We have taken important steps towards extending linear Dyna-style planning to multi-step planning. Multi-step linear Dyna-style planning uses multi-step linear models to project a simulated feature multiple steps into the future. For control, we proposed a different way of doing linear Dyna-style planning, that builds a model from state-action pair to state-action pair, and tracks the greedy action selection. Experimental results show that multi-step linear Dyna-style planning leads to better performance than existing single-step linear Dyna-style planning on Boyan chain and Mountain-car problems. Our experimental results show that linear Dyna-style planning can achieve a better performance by using different step-sizes for learning, modeling, and planning than using a uniform step-size for the three sub-procedures. While it is not clear from previous work, our results fully demonstrate the advantages of linear Dyna over TD/Q-learning for both policy evaluation and control.

Our work also sheds light on why previous attempts on developing independent online LSTD control were not successful (e.g., forgetting strategies (Sutton et al., 2008)). LSTD and Dyna-Q($\infty$) can become unstable because they magnify the model errors by looking infinite steps into the future. Current experiments do not include comparisons with any other LSTD control algorithm because we did not find in the literature an independent LSTD control algorithm. LSPI is usually off-line, and its extension to online control has to deal with online exploration (Li et al., 2009). Some researchers have combined LSTD in critic within the Actor-Critic framework (Xu et al., 2002; Peters & Schaal, 2008); however, LSTD there is still not an independent control algorithm.

## Acknowledgements

The authors received many feedbacks from Dr. Rich Sutton and Dr. Csaba Szepesvári. We gratefully acknowledge their help in improving the paper in many aspects. We also thank Alborz Geramifard for sending us Matlab code of tile coding. This research was supported by iCORE, NSERC and the Alberta Ingenuity Fund.

## Footnotes

[1]This is because $\gamma F^\pi$ has a spectral radius smaller than one, cf. Lemma 9.2.2 of (Bertsekas, Borkar & Nedich, 2004).

[2]Similarly $f^{(k)}$ can be obtained by shifting from $f^{(\infty)}$ by an amount that shrinks itself.

# References

Bertsekas, D. P., Borkar, V., & Nedich, A. (2004). Improved temporal difference methods with linear function approximation. *Learning and Approximate Dynamic Programming* (pp. 231–255). IEEE Press.

Boyan, J. A. (1999). Least-squares temporal difference learning. *ICML-16*.

Bradtke, S., & Barto, A. G. (1996). Linear least-squares algorithms for temporal difference learning. *Machine Learning*, *22*, 33–57.

Li, L., Littman, M. L., & Mansley, C. R. (2009). Online exploration in least-squares policy iteration. *AAMAS-8*.

Peters, J., & Schaal, S. (2008). Natural actor-critic. *Neurocomputing*, *71*, 1180–1190.

Sutton, R. S. (1990). Integrated architectures for learning, planning, and reacting based on approximating dynamic programming. *ICML-7*.

Sutton, R. S. (1995). TD models: modeling the world at a mixture of time scales. *ICML-12*.

Sutton, R. S., & Barto, A. G. (1998). *Reinforcement learning: An introduction*. MIT Press.

Sutton, R. S., Szepesvári, C., Geramifard, A., & Bowling, M. (2008). Dyna-style planning with linear function approximation and prioritized sweeping. *UAI-24*.

Tsitsiklis, J. N., & Van Roy, B. (1997). An analysis of temporal-difference learning with function approximation. *IEEE Transactions on Automatic Control*, *42*, 674–690.

Xu, X., He, H., & Hu, D. (2002). Efficient reinforcement learning using recursive least-squares methods. *Journal of Artificial Intelligence Research*, *16*, 259–292.

